# From Lasso regression to Feature vector machine

**Fan Li[1], Yiming Yang[1] and Eric P. Xing[1,2]**
[1] LTI and [2]CALD, School of Computer Science, Carnegie Mellon University,
Pittsburgh, PA USA 15213
{hustlf,yiming,epxing}@cs.cmu.edu

## Abstract

Lasso regression tends to assign zero weights to most irrelevant or redundant features, and hence is a promising technique for feature selection. Its limitation, however, is that it only offers solutions to linear models. Kernel machines with feature scaling techniques have been studied for feature selection with non-linear models. However, such approaches require to solve hard non-convex optimization problems. This paper proposes a new approach named the Feature Vector Machine (FVM). It reformulates the standard Lasso regression into a form isomorphic to SVM, and this form can be easily extended for feature selection with non-linear models by introducing kernels defined on feature vectors. FVM generates sparse solutions in the nonlinear feature space and it is much more tractable compared to feature scaling kernel machines. Our experiments with FVM on simulated data show encouraging results in identifying the small number of dominating features that are non-linearly correlated to the response, a task the standard Lasso fails to complete.

## 1  Introduction

Finding a small subset of most predictive features in a high dimensional feature space is an interesting problem with many important applications, e.g. in bioinformatics for the study of the genome and the proteome, and in pharmacology for high throughput drug screening.

Lasso regression ([Tibshirani *et al.*, 1996]) is often an effective technique for shrinkage and feature selection. The loss function of Lasso regression is defined as:

$$L = \sum_i (y_i - \sum_p \beta_p x_{ip})^2 + \lambda \sum_p ||\beta_p||_1$$

where $x_{ip}$ denotes the $p$th predictor (feature) in the $i$th datum, $y_i$ denotes the value of the response in this datum, and $\beta_p$ denotes the regression coefficient of the $p$th feature. The norm-1 regularizer $\sum_p ||\beta_p||_1$ in Lasso regression typically leads to a sparse solution in the feature space, which means that the regression coefficients for most irrelevant or redundant features are shrunk to zero. Theoretical analysis in [Ng *et al.*, 2003] indicates that Lasso regression is particularly effective when there are many irrelevant features and only a few training examples.

One of the limitations of standard Lasso regression is its assumption of linearity in the feature space. Hence it is inadequate to capture non-linear dependencies from features to responses (output variables). To address this limitation, [Roth, 2004] proposed "generalized Lasso regressions" (GLR) by introducing kernels. In GLR, the loss function is defined as

$$L = \sum_i (y_i - \sum_j \alpha_j k(x_i, \, x_j))^2 + \lambda \sum_i ||\alpha_i||_1$$

where $\alpha_j$ can be regarded as the regression coefficient corresponding to the $j$th basis in an *instance space* (more precisely, a kernel space with its basis defined on all examples), and $k(x_i, \, x_j)$ represents some kernel function over the "argument" instance $x_i$ and the "basis" instance $x_j$. The non-linearity can be captured by a non-linear kernel. This loss function typically yields a sparse solution in the instance space, but not in feature space where data was originally represented. Thus GLR does not lead to compression of data in the feature space.

[Weston *et al.*, 2000], [Canu *et al.*, 2002] and [Krishnapuram *et al.*, 2003] addressed the limitation from a different angle. They introduced *feature scaling kernels* in the form of:

$$K_\theta(x_i, x_j) = \phi(x_i * \theta)\phi(x_j * \theta) = K(x_i * \theta, x_j * \theta)$$

where $x_i * \theta$ denotes the component-wise product between two vectors: $x_i * \theta = (x_{i1}\theta_1, ..., x_{ip}\theta_p)$. For example, [Krishnapuram *et al.*, 2003] used a feature scaling polynomial kernel:

$$K_\gamma(x_i, x_j) = (1 + \sum_p \gamma_p x_{ip} x_{jp})^k,$$

where $\gamma_p = \theta_p^2$. With a norm-1 or norm-0 penalizer on $\gamma$ in the loss function of a feature scaling kernel machine, a sparse solution is supposed to identify the most influential features. Notice that in this formalism the feature scaling vector $\theta$ is inside the kernel function, which means that the solution space of $\theta$ could be non-convex. Thus, estimating $\theta$ in feature scaling kernel machines is a much harder problem than the convex optimization problem in conventional SVM of which the weight parameters to be estimated are outside of the kernel functions.

What we are seeking for here is an alternative approach that guarantees a sparse solution in the feature space, that is sufficient for capturing both linear and non-linear relationships between features and the response variable, and that does not involve parameter optimization inside of kernel functions. The last property is particularly desirable in the sense that it will allow us to leverage many existing works in kernel machines which have been very successful in SVM-related research.

We propose a new approach where the key idea is to re-formulate and extend Lasso regression into a form that is similar to SVM except that it generates a sparse solution in the feature space rather than in the instance space. We call our newly formulated and extended Lasso regression "Feature Vector Machine" (FVM). We will show (in Section 2) that FVM has many interesting properties that mirror SVM. The concepts of support vectors, kernels and slack variables can be easily adapted in FVM. Most importantly, all the parameters we need to estimate for FVM are outside of the kernel functions, ensuring the convexity of the solution space, which is the same as in SVM. [1] When a linear kernel is put to use with no slack variables, FVM reduces to the standard Lasso regression.

We notice that [Hochreiter *et al.*, 2004] has recently developed an interesting feature selection technique named "potential SVM", which has the same form as the basic version of FVM (with linear kernel and no slack variables). However, they did not explore the relationship between "potential SVM" and Lasso regression. Furthermore, their method does not work for feature selection tasks with non-linear models since they did not introduce the concepts of kernels defined on feature vectors.

In section 2, we analyze some geometric similarities between the solution hyper-planes in the standard Lasso regression and in SVM. In section 3, we re-formulate Lasso regression in a SVM style form. In this form, all the operations on the training data can be expressed by dot products between feature vectors. In section 4, we introduce kernels (defined for feature vectors) to FVM so that it can be used for feature selection with non-linear models. In section 5, we give some discussions on FVM. In section 6, we conduct experiments and in section 7 we give conclusions.

## 2 Geometric parity between the solution hyper-planes of Lasso regression and SVM

Formally, let $\mathbf{X} = [x_1, \ldots, x_N]$ denote a sample matrix, where each column $x_i = (x_1, \ldots, x_K)^T$ represents a *sample vector* defined on $K$ features. A *feature vector* can be defined as a transposed row in the sample matrix, i.e., $f_q = (x_{1q}, \ldots, x_{Nq})^T$ (corresponding to the $q$ row of $\mathbf{X}$). Note that we can write $\mathbf{X}^T = [f_1, \ldots, f_K] = \mathbf{F}$. For convenience, let $y = (y_1, \ldots, y_n)^T$ denote a *response vector* containing the responses corresponding to all the samples.

Now consider an *example space* of which each basis is represented by an $x_i$ in our sample matrix (note that this is different from the space "spanned" by the sample vectors). Under the example space, both the features $f_q$ and the response vector $y$ can be regarded as a point in this space. It can be shown that the solution of Lasso regression has a very intuitive meaning in the example space: the regression coefficients can be regarded as the weights of feature vectors in the example space; moreover, all the non-zero weighted feature vectors are on two parallel hyper-planes in the example space. These feature vectors, together with the response variable, determine the directions of these two hyper-planes. This geometric view can be drawn from the following recast of the Lasso regression due to [Perkins *et al.*, 2003]:

$$|\sum_i (y_i - \sum_p \beta_p x_{ip}) x_{iq}| \leq \frac{\lambda}{2}, \quad \forall q$$

$$\Rightarrow \quad |f_q(y - [f_1, \ldots, f_K]\beta)| \leq \frac{\lambda}{2}, \quad \forall q. \tag{1}$$

It is apparent from the above equation that $y - [f_1, \ldots, f_K]\beta$ defines the orientation of a separation hyper-plane. It can be shown that equality only holds for non-zero weighted features, and all the zero weighted feature vectors are between the hyper-planes with $\lambda/2$ margin (Fig. 1a).

The separating hyper-planes due to (hard, linear) SVM have similar properties as those of the regression hyper-planes described above, although the former are now defined in the feature space (in which each axis represents a feature and each point represents a sample) instead of the example space. In an SVM, all the non-zero weighted samples are also on the two $\lambda/2$-margin separating hyper-planes (as is the case in Lasso regression), whereas all the zero-weighted samples are now *outside* the pair of hyper-planes (Fig 1b). It's well known that the classification hyper-planes in SVM can be extended to hyper-surfaces by introducing kernels defined for *example vectors*. In this way, SVM can model non-linear dependencies between samples and the classification boundary. Given the similarity of the

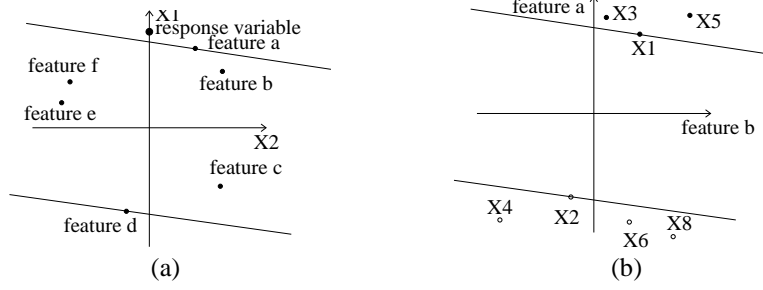

Figure 1: Lasso regression vs. SVM. (a) The solution of Lasso regression in the example space. $X1$ and $X2$ represent two examples. Only feature $a$ and $d$ have non-zero weights, and hence the *support features*. (b)The solution of SVM in the feature space. Sample $X1$, $X3$ and $X5$ are in one class and $X2$, $X4$, $X6$ and $X8$ are in the other. $X1$ and $X2$ are the *support vectors* (i.e., with non-zero weights).

geometric structures of Lasso regression and SVM, it is nature to pursue in parallel how one can apply similar "kernel tricks" to the *feature vectors* in Lasso regression, so that its feature selection power can be extended to non-linear models. This is the intension of this paper, and we envisage full leverage of much of the computational/optimization techniques well-developed in the SVM community in our task.

## 3   A re-formulation of Lasso regression akin to SVM

[Hochreiter *et al.*, 2004] have proposed a "potential SVM" as follows:

$$
\begin{cases}
\underline{\min}_\beta & \frac{1}{2}\sum_i(\sum_p \beta_p x_{ip})^2 \\
\underline{\text{s.t.}} & |\sum_i(y_i - \sum_p \beta_p x_{ip})x_{iq}| \leq \frac{\lambda}{2} \qquad \forall q.
\end{cases}
\tag{2}
$$

To clean up a little bit, we rewrite Eq. (2) in linear algebra format:

$$
\begin{cases}
\underline{\min}_\beta & \frac{1}{2}\|[f_1^T,\ldots,f_K^T]\beta\|^2 \\
\underline{\text{s.t.}} & |f_q(y - [f_1,\ldots,f_K]\beta)| \leq \frac{\lambda}{2}, \quad \forall q.
\end{cases}
\tag{3}
$$

A quick eyeballing of this formulation reveals that it shares the same constrain function needed to be satisfied in Lasso regression. Unfortunately, this connection was not further explored in [Hochreiter *et al.*, 2004], e.g., to relate the objection function to that of the Lasso regression, and to extend the objective function using kernel tricks in a way similar to SVM. Here we show that the solution to Eq. (2) is exactly the same as that of a standard Lasso regression. In other words, Lasso regression can be re-formulated as Eq. (2). Then, based on this re-formulation, we show how to introduce kernels to allow feature selection under a non-linear Lasso regression. We refer to the optimization problem defined by Eq. (3), and its kernelized extensions, as *feature vector machine* (FVM).

**Proposition 1:** For a Lasso regression problem $min_\beta \sum_i(\sum_p x_{ip}\beta_p - y_i)^2 + \lambda\sum_p |\beta_p|$, if we have $\beta$ such that: if $\beta_q = 0$, then $|\sum_i(\sum_p \beta_p x_{ip} - y_i)x_{iq}| < \frac{\lambda}{2}$; if $\beta_q < 0$, then $\sum_i(\sum_p \beta_p x_{ip} - y_i)x_{iq} = \frac{\lambda}{2}$; and if $\beta_q > 0$, then $\sum_i(\sum_p \beta_p x_{ip} - y_i)x_{iq} = -\frac{\lambda}{2}$, then $\beta$ is the solution of the Lasso regression defined above. For convenience, we refer to the aforementioned three conditions on $\beta$ as the *Lasso sandwich*.

**Proof:** see [Perkins *et al.*, 2003]. ∎

**Proposition 2:** For Problem (3), its solution $\beta$ satisfies the *Lasso sandwich*

**Sketch of proof:** Following the equivalence between feature matrix $\mathbf{F}$ and sample matrix $\mathbf{X}$ (see the begin of §2), Problem (3) can be re-written as:

$$\begin{cases} \underline{\min}_\beta & \frac{1}{2}||X^T\beta||^2 \\ \underline{\text{s.t.}} & X(X^T\beta - y) - \frac{\lambda}{2}e \leq 0 \\ & X(X^T\beta - y) + \frac{\lambda}{2}e \geq 0 \end{cases}, \qquad (4)$$

where $e$ is a one-vector of $K$ dimensions. Following the standard constrained optimization procedure, we can derive the dual of this optimization problem. The Lagrange L is given by

$$L = \frac{1}{2}\beta^T XX^T\beta - \alpha_+^T(X(X^T\beta - y) + \frac{\lambda}{2}e) + \alpha_-^T(X(X^T\beta - y) + \frac{\lambda}{2}e)$$

where $\alpha_+$ and $\alpha_-$ are $K \times 1$ vectors with positive elements. The optimizer satisfies:

$$\nabla_\beta L = XX^T\beta - XX^T(\alpha_+ - \alpha_-) = 0$$

Suppose the data matrix X has been pre-processed so that the feature vectors are centered and normalized. In this case the elements of $XX^T$ reflect the correlation coefficients of feature pairs and $XX^T$ is non-singular. Thus we know $\beta = \alpha_+ - \alpha_-$ is the solution of this loss function. For any element $\beta_q > 0$, obviously $\alpha_{+q}$ should be larger than zero. From the KKT condition, we know $\sum_i(y_i - \sum_p \beta_p x_{ip})x_{iq} = -\frac{\lambda}{2}$ holds at this time. For the same reason we can get when $\beta_q < 0$, $\alpha_{-q}$ should be larger than zero thus $\sum_i(y_i - \sum_p \beta_p x_{ip})x_{iq} = \frac{\lambda}{2}$ holds. When $\beta_q = 0$, $\alpha_{+q}$ and $\alpha_{-q}$ must both be zero (it's easy to see they can not be both non-zero from KKT condition), thus from KKT condition, both $\sum_i(y_i - \sum_p \beta_p x_{ip})x_{iq} > -\frac{\lambda}{2}$ and $\sum_i(y_i - \sum_p \beta_p x_{ip})x_{iq} < \frac{\lambda}{2}$ hold now, which means $|\sum_i(y_i - \sum_p \beta_p x_{ip})x_{iq}| < \frac{\lambda}{2}$ at this time. ∎

**Theorem 3:** Problem (3) ≡ Lasso regression.

**Proof.** Follows from proposition 1 and proposition 2. ∎

## 4 Feature kernels

In many cases, the dependencies between feature vectors are non-linear. Analogous to the SVM, here we introduce kernels that capture such non-linearity. Note that unlike SVM, our kernels are defined on feature vectors instead of the sampled vectors (i.e., the rows rather than the columns in the data matrix). Such kernels can also allow us to easily incorporate certain domain knowledge into the classifier.

Suppose that two feature vectors $f_p$ and $f_q$ have a non-linear dependency relationship. In the absence of linear interaction between $f_p$ and $f_q$ in the the original space, we assume that they can be mapped to some (higher dimensional, possibly infinite-dimensional) space via transformation $\phi(\cdot)$, so that $\phi(f_q)$ and $\phi(f_q)$ interact linearly, i.e., via a dot product $\phi(f_p)^T\phi(f_q)$. We introduce kernel $K(f_q, f_p) = \phi(f_p)^T\phi(f_q)$ to represent the outcome of this operation.

Replacing $f$ with $\phi(f)$ in Problem (3), we have

$$\begin{cases} \underline{\min}_\beta & \frac{1}{2}\sum_{p,q}\beta_p\beta_q K(f_p, f_p) \\ \underline{\text{s.t.}} & \forall q, \quad |\sum_p \beta_p K(f_q, f_p) - K(f_q, y)| \leq \frac{\lambda}{2} \end{cases} \qquad (5)$$

Now, in Problem 5, we no longer have $\phi(\cdot)$, which means we do not have to work in the transformed feature space, which could be high or infinite dimensional, to capture non-linearity of features. The kernel $K(\cdot, \cdot)$ can be any symmetric semi-positive definite matrix.

When domain knowledge from experts is available, it can be incorporated into the choice of kernel (e.g., based on the distribution of feature values). When domain knowledge is not available, we can use some general kernels that can detect non-linear dependencies without any distribution assumptions. In the following we give one such example.

One possible kernel is the mutual information [Cover *et al.*, 1991] between two feature vectors: $K(f_p, f_q) = MI(f_p, f_q)$. This kernel requires a pre-processing step to discritize the elements of features vectors because they are continuous in general. In this paper, we discritize the continuous variables according to their ranks in different examples. Suppose we have $N$ examples in total. Then for each feature, we sort its values in these $N$ examples. The first $m$ values (the smallest $m$ values) are assigned a scale 1. The $m + 1$ to $2m$ values are assigned a scale 2. This process is iterated until all the values are assigned with corresponding scales. It's easy to see that in this way, we can guarantee that for any two features $p$ and $q$, $K(f_p, f_p) = K(f_q, f_q)$, which means the feature vectors are normalized and have the same length in the $\phi$ space (residing on a unit sphere centered at the origin).

Mutual information kernels have several good properties. For example, it is symmetric (i.e., $K(f_p, f_q) = K(f_q, f_p)$, non-negative, and can be normalized. It also has intuitive interpretation related to the redundancy between features. Therefore, a non-linear feature selection using generalized Lasso regression with this kernel yields human interpretable results.

## 5   Some extensions and discussions about FVM

As we have shown, FVM is a straightforward feature selection algorithm for nonlinear features captured in a kernel; and the selection can be easily done by solving a standard SVM problem in the feature space, which yield an optimal vector $\beta$ of which most elements are zero. It turns out that the same procedure also seemlessly leads to a Lasso-style regularized nonlinear regression capable of predicting the response given data in the original space.

In the prediction phase, all we have to do is to keep the trained $\beta$ fixed, and turn the optimization problem (5) into an analogous one that optimizes over the response $y$. Specifically, given a new sample $x_t$ of unknown response, our sample matrix $\mathbf{X}$ grows by one column $\mathbf{X} \rightarrow [\mathbf{X}, x_t]$, which means all our feature vectors gets one more dimension. We denote the newly elongated features by $F' = \{f'_q\}_{q \in A}$ (note that $A$ is the pruned index set corresponding to features whose weight $\beta_q$ is non-zero). Let $y'$ denote the elongated response vector due to the newly given sample: $y' = (y_1, ..., y_N, y_t)^T$, it can be shown that the optimum response $y_t$ can be obtained by solving the following optimization problem [2]:

$$\min_{y_t} K(y', y') - 2 \sum_{p \in A} \beta_p K(y', f'_p) \qquad (6)$$

When we replace the kernel function $K$ with a linear dot product, FVM reduces to Lasso regression. Indeed, in this special case, it is easy to see from Eq. (6) that $y_t = \sum_{p \in A} \beta_p x_{tp}$, which is exactly how Lasso regression would predict the response. In this case one predicts $y_t$ according to $\beta$ and $x_t$ without using the training data $\mathbf{X}$. However, when a more complex kernel is used, solving Eq. (6) is not always trivial. In general, to predict $y_t$, we need not only $x_t$ and $\beta$, but also the non-zero weight features extracted from the training data.

$$min_\beta ||\phi(y') - \sum_p \beta_p \phi(f'_p)||^2 + \sum_p ||\beta_p||_1.$$

Replacing the opt. argument $\beta$ with $y$ and dropping terms irrelevant to $y_t$, we will arrive at Eq. (6).

As in SVM, we can introduce slack variables into FVM to define a "soft" feature surface. But due to space limitation, we omit details here. Essentially, most of the methodologies developed for SVM can be easily adapted to FVM for nonlinear feature selection.

## 6 Experiments

We test FVM on a simulated dataset with 100 features and 500 examples. The response variable $y$ in the simulated data is generated by a highly nonlinear rule:

$$ y = sin(10 * f_1 - 5) + 4 * \sqrt{1 - f_2^2} - 3 * f_3 + \xi. $$

Here feature $f_1$ and $f_3$ are random variables following a uniform distribution in $[0, 1]$; feature $f_2$ is a random variable uniformly distributed in $[-1, 1]$; and $\xi$ represents Gaussian noise. The other 97 features $f_4, f_5, ..., f_{100}$ are conditionally independent of $y$ given the three features $f_1$, $f_2$ and $f_3$. In particular, $f_4, ..., f_{33}$ are all generated by the rule $f_j = 3 * f_1 + \xi$; $f_{34}, ..., f_{72}$ are all generated by the rule $f_j = sin(10 * f_2) + \xi$; and the remaining features ($f_{73}, ..., f_{100}$) simply follow a uniform distribution in $[0, 1]$. Fig. 2 shows our data projected in a space spanned by $f_1$ and $f_2$ and $y$.

We use a mutual information kernel for our FVM. For each feature, we sort its value in different examples and use the rank to discritize these values into 10 scales (thus each scale corresponds to 50 data points). An FVM can be solved by quadratic programming, but more efficient solutions exist. [Perkins *et al.*, 2003] has proposed a fast grafting algorithm to solve Lasso regression, which is a special case of FVM when linear kernel is used. In our implementation, we extend the idea of fast grafting algorithm to FVM with more general kernels. The only difference is that, each time when we need to calculate $\sum_i x_{pi} x_{qi}$, we calculate $K(f_p, f_q)$ instead. We found that fast grafting algorithm is very efficient in our case because it uses the sparse property of the solution of FVM.

We apply both standard Lasso regression and FVM with mutual information kernel on this dataset. The value of the regularization parameter $\lambda$ can be tuned to control the number of non-zero weighted features. In our experiment, we tried two choices of the $\lambda$, for both FVM and the standard Lasso regression. In one case, we set $\lambda$ such that only 3 non-zero weighted features are selected; in another case, we relaxed a bit and allowed 10 features.

The results are very encouraging. As shown in Fig. (3), under stringent $\lambda$, FVM successfully identified the three correct features, $f_1$, $f_2$ and $f_3$, whereas Lasso regression has missed $f_1$ and $f_2$, which are non-linearly correlated with $y$. Even when $\lambda$ was relaxed, Lasso regression still missed the right features, whereas FVM was very robust.

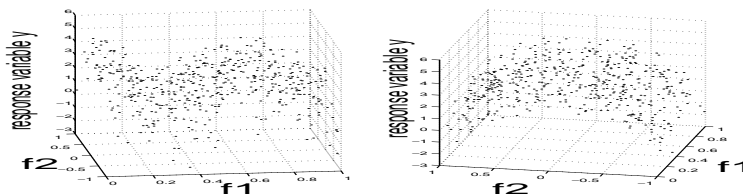

Figure 2: The responses $y$ and the two features $f_1$ and $f_2$ in our simulated data. Two graphs from different angles are plotted to show the distribution more clearly in 3D space.

## 7 Conclusions

In this paper, we proposed a novel non-linear feature selection approach named FVM, which extends standard Lasso regression by introducing kernels on feature vectors. FVM

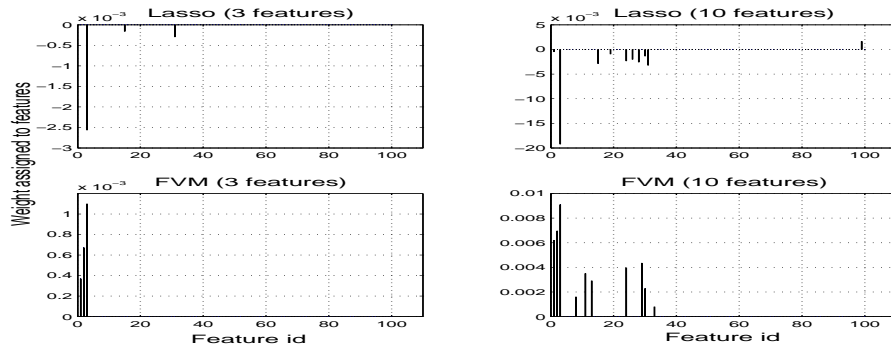

Figure 3: Results of FVM and the standard Lasso regression on this dataset. The X axis represents the feature IDs and the Y axis represents the weights assigned to features. The two left graphs show the case when 3 features are selected by each algorithm and the two right graphs show the case when 10 features are selected. From the down left graph, we can see that FVM successfully identified $f_1, f_2$ and $f_3$ as the three non-zero weighted features. From the up left graph, we can see that Lasso regression missed $f_1$ and $f_2$, which are non-linearly correlated with $y$. The two right graphs show similar patterns.

has many interesting properties that mirror the well-known SVM, and can therefore leverage many computational advantages of the latter approach. Our experiments with FVM on highly nonlinear and noisy simulated data show encouraging results, in which it can correctly identify the small number of dominating features that are non-linearly correlated to the response variable, a task the standard Lasso fails to complete.

## Footnotes

[1]Notice that we can not only use FVM to select important features from training data, but also use it to predict the values of response variables for test data (see section 5). We have shown that we only need convex optimization in the training phase of FVM. In the test phase, FVM makes a prediction for each test example independently. This only involves with a one-dimensional optimization problem with respect to the response variable for the test example. Although the optimization in the test phase may be non-convex, it will be relatively easy to solve because it is only one-dimensional. This is the price we pay for avoiding the high dimensional non-convex optimization in the training phase, which may involve thousands of model parameters.

[2]For simplicity we omit details here, but as a rough sketch, note that Eq. (5) can be reformed as

## References

[Canu *et al.*, 2002]  Canu, S. and Grandvalet, Y. Adaptive Scaling for Feature Selection in SVMs NIPS 15, 2002

[Hochreiter *et al.*, 2004]  Hochreiter, S. and Obermayer, K. Gene Selection for Microarray Data. In Kernel Methods in Computational Biology, pp. 319-355, MIT Press, 2004.

[Krishnapuram *et al.*, 2003]  Krishnapuram, B. et al. Joint classifier and feature optimization for cancer diagnosis using gene expression data. The Seventh Annual International Conference on Research in Computational Molecular Biology (RECOMB) 2003, ACM press, April 2003

[Ng *et al.*, 2003]  Ng, A. Feature selection, L1 vs L2 regularization, and rotational invariance. ICML 2004

[Perkins *et al.*, 2003]  Perkins, S., Lacker, K. & Theiler, J. Grafting: Fast,Incremental Feature Selection by gradient descent in function space JMLR 2003 1333-1356

[Roth, 2004]  Roth, V. The Generalized LASSO. IEEE Transactions on Neural Networks (2004), Vol. 15, NO. 1.

[Tibshirani *et al.*, 1996]  Tibshirani, R. Optimal Reinsertion:Regression shrinkage and selection via the lasso. J.R.Statist. Soc. B(1996), 58,No.1, 267-288

[Cover *et al.*, 1991]  Cover, TM. and Thomas, JA. Elements in Information Theory. New York: John Wiley & Sons Inc (1991).

[Weston *et al.*, 2000]  Weston, J., Mukherjee, S., Chapelle, O., Pontil, M., Poggio, T. and Vapnik V. Feature Selection for SVMs NIPS 13, 2000
